# A Connectionist Learning Approach to Analyzing Linguistic Stress

**Prahlad Gupta**
Department of Psychology
Carnegie Mellon University
Pittsburgh, PA 15213

**David S. Touretzky**
School of Computer Science
Carnegie Mellon University
Pittsburgh, PA 15213

## Abstract

We use connectionist modeling to develop an analysis of stress systems in terms of ease of learnability. In traditional linguistic analyses, learnability arguments determine default parameter settings based on the feasibilty of logically deducing correct settings from an initial state. Our approach provides an empirical alternative to such arguments. Based on perceptron learning experiments using data from nineteen human languages, we develop a novel characterization of stress patterns in terms of six parameters. These provide both a partial description of the stress pattern itself and a prediction of its learnability, without invoking abstract theoretical constructs such as metrical feet. This work demonstrates that machine learning methods can provide a fresh approach to understanding linguistic phenomena.

## 1 LINGUISTIC STRESS

The domain of stress systems in language is considered to have a relatively good linguistic theory, called *metrical phonology*[1]. In this theory, the stress patterns of many languages can be described concisely, and characterized in terms of a set of linguistic "parameters," such as bounded vs. unbounded metrical feet, left vs. right dominant feet, etc.[2] In many languages, stress tends to be placed on certain kinds of syllables rather than on others; the former are termed *heavy* syllables, and the latter *light* syllables. Languages that distinguish

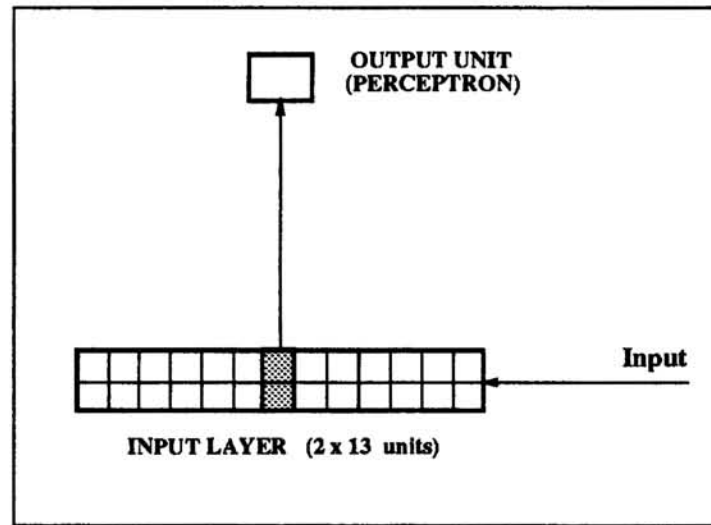

Figure 1: Perceptron model used in simulations.

between heavy and light syllables are termed *quantity-sensitive* (QS), while languages that do not make this distinction are termed *quantity-insensitive* (QI). In some QS languages, what counts as a heavy syllable is a closed syllable (a syllable that ends in a consonant), while in others it is a syllable with a long vowel. We examined the stress patterns of nineteen QI and QS systems, summarized and exemplified in Table 1. The data were drawn primarily from descriptions in [Hayes 80].

## 2   PERCEPTRON SIMULATIONS

In separate experiments, we trained a perceptron to produce the stress pattern of each of these languages. Two input representations were used. In the *syllabic* representation, used for QI patterns only, a syllable was represented as a [1 1] vector, and [0 0] represented no syllable. In the *weight-string* representation, which was necessary for QS languages, the input patterns used were [1 0] for a heavy syllable, [0 1] for a light syllable, and [0 0] for no syllable. For stress systems with up to two levels of stress, the output targets used in training were 1.0 for primary stress, 0.5 for secondary stress, and 0 for no stress. For stress systems with three levels of stress, the output targets were 0.6 for secondary stress, 0.35 for tertiary stress, and 1.0 and 0 respectively for primary stress and no stress. The input data set for all stress systems consisted of all word-forms of up to seven syllables. With the syllabic input representation there are 7 of these, and with the weight-string representation, there are 255. The perceptron's input array was a buffer of 13 syllables; each word was processed one syllable at a time by sliding it through the buffer (see Figure 1). The desired output at each step was the stress level of the middle syllable of the buffer. Connection weights were adjusted at each step using the back-propagation learning algorithm [Rumelhart 86]. One *epoch* consisted of one presentation of the entire training set. The network was trained for as many epochs as necessary to ensure that the stress value produced by the perceptron was within 0.1 of the target value, for each syllable of the word, for all words in the training set. A *learning rate* of 0.05 and *momentum* of 0.90 was used in all simulations. Initial weights were uniformly distributed random values in the range ±0.5. Each simulation was run at least three times, and the learning times averaged.

| REF | LANGUAGE | DESCRIPTION OF STRESS PATTERN | EXAMPLES |
|---|---|---|---|
| Quantity-Insensitive Languages: | | | |
| L1 | Latvian | Fixed word-initial stress. | $S^1S^0S^0S^0S^0S^0S^0$ |
| L2 | French | Fixed word-final stress. | $S^0S^0S^0S^0S^0S^0S^1$ |
| L3 | Maranungku | Primary stress on first syllable, secondary stress on alternate succeeding syllables. | $S^1S^0S^2S^0S^2S^0S^2$ |
| L4 | Weri | Primary stress on last syllable, secondary stress on alternate preceding syllables. | $S^2S^0S^2S^0S^2S^0S^1$ |
| L5 | Garawa | Primary stress on first syllable, secondary stress on penultimate syllable, tertiary stress on alternate syllables preceding the penult, no stress on second syllable. | $S^1S^0S^0S^3S^0S^2S^0$ |
| L6 | Lakota | Primary stress on second syllable. | $S^0S^1S^0S^0S^0S^0S^0$ |
| L7 | Swahili | Primary stress on penultimate syllable. | $S^0S^0S^0S^0S^0S^1S^0$ |
| L8 | Paiute | Primary stress on second syllable, secondary stress on alternate succeeding syllables. | $S^0S^1S^0S^2S^0S^2S^0$ |
| L9 | Warao | Primary stress on penultimate syllable, secondary stress on alternate preceding syllables. | $S^0S^2S^0S^2S^0S^1S^0$ |
| Quantity-Sensitive Languages: | | | |
| L10 | Koya | Primary stress on first syllable, secondary stress on heavy syllables. (Heavy = closed syllable or syllable with long vowel.) | $L^1L^0L^0H^2L^0L^0L^0$ $L^1L^0L^0L^0L^0L^0L^0$ |
| L11 | Eskimo | (Primary) stress on final and heavy syllables. (Heavy = closed syllable.) | $L^0L^0L^0H^1L^0L^0L^1$ $L^0L^0L^0L^0L^0L^0L^1$ |
| L12 | Gurkhali | Primary stress on first syllable except when first syllable light and second syllable heavy. (Heavy = long vowel.) | $L^1L^0L^0H^0L^0L^0L^0$ $L^0H^1L^0H^0L^0L^0L^0$ |
| L13 | Yapese | Primary stress on last syllable except when last is light and penultimate heavy. (Heavy = long vowel.) | $L^0L^0L^0H^0L^0L^0L^1$ $L^0H^0L^0H^0L^0H^1L^0$ |
| L14 | Ossetic | Primary stress on first syllable if heavy, else on second syllable. (Heavy = long vowel.) | $H^1L^0L^0H^0L^0L^0L^0$ $L^0L^1L^0L^0L^0L^0L^0$ |
| L15 | Rotuman | Primary stress on last syllable if heavy, else on penultimate syllable. (Heavy = long vowel.) | $L^0L^0L^0H^0L^0L^0H^1$ $L^0L^0L^0L^0L^0L^1L^0$ |
| L16 | Komi | Primary stress on first heavy syllable, or on last syllable if none heavy. (Heavy = long vowel.) | $L^0L^0H^1L^0L^0H^0L^0$ $L^0L^0L^0L^0L^0L^0L^1$ |
| L17 | Cheremis | Primary stress on last heavy syllable, or on first syllable if none heavy. (Heavy = long vowel.) | $L^0L^0H^0L^0L^0H^1L^0$ $L^1L^0L^0L^0L^0L^0L^0$ |
| L18 | Mongolian | Primary stress on first heavy syllable, or on first syllable if none heavy. (Heavy = long vowel.) | $L^0L^0H^1L^0L^0H^0L^0$ $L^1L^0L^0L^0L^0L^0L^0$ |
| L19 | Mayan | Primary stress on last heavy syllable, or on last syllable if none heavy. (Heavy = long vowel.) | $L^0L^0H^0L^0L^0H^1L^0$ $L^0L^0L^0L^0L^0L^0L^1$ |

Table 1: Stress patterns: description and example stress assignment. Examples are of stress assignment in seven-syllable words. Primary stress is denoted by the superscript 1 (e.g., $S^1$), secondary stress by the superscript 2, tertiary stress by the superscript 3, and no stress by the superscript 0. "S" indicates an arbitrary syllable, and is used for the QI stress patterns. For QS stress patterns, "H" and "L" are used to denote Heavy and Light syllables, respectively.

## 3   PRELIMINARY ANALYSIS OF LEARNABILITY OF STRESS

The learning times differ considerably for {Latvian, French}, {Maranungku, Weri}, {Lakota, Polish} and Garawa, as shown in the last column of Table 2. Moreover, Paiute and Warao were unlearnable with this model.[3] Differences in learning times for the various stress patterns suggested that the factors ("parameters") listed below are relevant in determining learnability.

1. **Inconsistent Primary Stress (IPS):** it is computationally expensive to learn the pattern if neither edge receives primary stress except in mono- and di-syllables; this can be regarded as an index of computational complexity that takes the values {0, 1}: 1 if an edge receives primary stress inconsistently, and 0, otherwise.

2. **Stress clash avoidance (SCA):** if the components of a stress pattern can potentially lead to *stress clash*[4], then the language may either actually permit such stress clash, or it may avoid it. This index takes the values {0, 1}: 0 if stress clash is permitted, and 1 if stress clash is avoided.

3. **Alternation (Alt):** an index of learnability with value 0 if there is no alternation, and value 1 if there is. Alternation refers to a stress pattern that repeats on alternate syllables.

4. **Multiple Primary Stresses (MPS):** has value 0 if there is exactly one primary stress, and value 1 if there is more then one primary stress. It has been assumed that a repeating pattern of primary stresses will be on alternate, rather than adjacent syllables. Thus, [Alternation=0] implies [MPS=0]. Some of the hypothetical stress patterns examined below include ones with more than one primary stress; however, as far as is known, no actually occurring QI stress pattern has more than one primary stress.

5. **Multiple Stress Levels (MSL):** has value 0 if there is a single level of stress (primary stress only), and value 1 otherwise.

Note that it is possible to order these factors with respect to each other to form a five-digit binary string characterizing the ease/difficulty of learning. That is, the computational complexity of learning a stress pattern can be characterized as a 5-bit binary number whose bits represent the five factors above, in decreasing order of significance. Table 2 shows that this characterization captures the learning times of the QI patterns quite accurately. As an example of how to read Table 2, note that Garawa takes longer to learn than Latvian (165 vs. 17 epochs). This is reflected in the parameter setting for Garawa, "01101", being lexicographically greater than that for Latvian, "00000". A further noteworthy point is that this framework provides an account of the non-learnability of Paiute and Warao, viz,. that stress patterns whose parameter string is lexicographically greater than "10000" are unlearnable by the perceptron.

## 4   TESTING THE QI LEARNABILITY PREDICTIONS

We devised a series of thirty artificial QI stress patterns (each a variation on some language in Table 1) to examine our parameter scheme in more detail. The details of the patterns

| IPS | SCA | Alt | MPS | MSL | QI LANGUAGES | REF | EPOCHS (syllabic) |
|---|---|---|---|---|---|---|---|
| 0 | 0 | 0 | 0 | 0 | Latvian<br>French | L1<br>L2 | 17<br>16 |
| 0 | 0 | 1 | 0 | 1 | Maranungku<br>Weri | L3<br>L4 | 37<br>34 |
| 0 | 1 | 1 | 0 | 1 | Garawa | L5 | 165 |
| 1 | 0 | 0 | 0 | 0 | Lakota<br>Swahili | L6<br>L7 | 255<br>254 |
| 1 | 0 | 1 | 0 | 1 | Paiute<br>Warao | L8<br>L9 | **<br>** |

Table 2: Preliminary analysis of learning times for QI stress systems, using the *syllabic* input representation. IPS=Inconsistent Primary Stress; SCA=Stress Clash Avoidance; Alt=Alternation; MPS=Multiple Primary Stresses; MSL=Multiple Stress Levels. References L1-L9 refer to Table 1.

| Agg | IPS | SCA | Alt | MPS | MSL | QI LANGS | REF | TIME | QS LANGS | REF | TIME |
|---|---|---|---|---|---|---|---|---|---|---|---|
| 0 | 0 | 0 | 0 | 0 | 0 | Latvian<br>French | L1<br>L2 | 2<br>2 | | | |
| 0 | 0 | 0 | 0 | 0 | 1 | | | | Koya | L10 | 2 |
| 0 | 0 | 0 | 0 | 1 | 0 | | | | Eskimo | L11 | 3 |
| 0 | 0 | 0 | 1 | 0 | 1 | Maranungku<br>Weri | L3<br>L4 | 3<br>3 | | | |
| 0 | 0 | 1 | 1 | 0 | 1 | Garawa | L5 | 7 | | | |
| 0 | 0.25 | 0 | 0 | 0 | 0 | | | | Gurkhali<br>Yapese | L12<br>L13 | 19<br>19 |
| 0 | 0.50 | 0 | 0 | 0 | 0 | | | | Ossetic<br>Rotuman | L14<br>L15 | 30<br>29 |
| 0 | 1 | 0 | 0 | 0 | 0 | Lakota<br>Swahili | L6<br>L7 | 10<br>10 | | | |
| 0 | 1 | 0 | 1 | 0 | 1 | Paiute<br>Warao | L8<br>L9 | **<br>** | | | |
| 1 | 0 | 0 | 0 | 0 | 0 | | | | Komi<br>Cheremis | L16<br>L17 | 216<br>212 |
| 2 | 0 | 0 | 0 | 0 | 0 | | | | Mongolian<br>Mayan | L18<br>L19 | 2306<br>2298 |

Table 3: Summary of results and analysis of QI and QS learning (using *weight-string* input representations). Agg=Aggregative Information; IPS=Inconsistent Primary Stress; SCA=Stress Clash Avoidance; Alt=Alternation; MPS=Multiple Primary Stresses; MSL=Multiple Stress Levels. References index into Table 1. Time is learning time in *epochs*.

are not crucial for present purposes (see [Gupta 92] for details). What is important to note is that the learnability predictions generated by the analytical scheme described in the previous section show good agreement with actual perceptron learning experiments on these patterns.

The learning results are summarized in Table 4. It can be seen that the 5-bit characterization fits the learning times of various actual and hypothetical patterns reasonably well (although there are exceptions – for example, the hypothetical stress patterns with reference numbers h21 through h25 have a higher 5-bit characterization than other stress patterns, but lower learning times.) Thus, the "complexity measure" suggested here appears to identify a number of factors relevant to the learnability of QI stress patterns within a minimal two-layer connectionist architecture. It also assesses their relative impacts. The analysis is undoubtedly a simplification, but it provides a completely novel framework within which to relate the various learning results. The important point to note is that this analytical framework arises from a consideration of (a) the nature of the stress systems, and (b) the learning results from simulations. That is, this framework is empirically based, and makes no reference to abstract constructs of the kind that linguistic theory employs. Nevertheless, it provides a descriptive framework, much as the linguistic theory does.

## 5    INCORPORATING QS SYSTEMS INTO THE ANALYSIS

Consideration of the QS stress patterns led to *refinement* of the IPS parameter without changing its setting for the QI patterns. This parameter is modified so that its value indicates *the proportion of cases in which primary stress is not assigned at the edge of a word*. Additionally, through analysis of connection weights for QS patterns, a sixth parameter, *Aggregative Information*, is added as a further index of computational complexity.

**6. Aggregative Information (Agg)** : has value 0 if no aggregative information is required (*single-positional* information suffices); 1 if one kind of aggregative information is required; and 2 if two kinds of aggregative information are required.

Detailed discussion of the analysis leading to these refinements is beyond the scope of this paper; the interested reader is referred to [Gupta 92]. The point we wish to make here is that, with these modifications, the same parameter scheme can be used for both the QI and QS language classes, with good learnability predictions *within* each class, as shown in Table 3. Note that in this table, learning times for all languages are reported in terms of the weight-string representation (255 input patterns) rather than the unweighted syllabic representation (7 input patterns) used for the initial QI studies. Both the QI and QS results fall into a single analysis within this generalized parameter scheme and weight-string representation, but with a less perfect fit than the within-class results.

## 6    DISCUSSION

Traditional linguistic analysis has devised abstract theoretical constructs such as "metrical foot" to *describe* linguistic stress systems. Learnability arguments were then used to determine default parameter settings (e.g., whether feet should by default be assumed to be bounded or unbounded, left or right dominant, etc.) based on the feasibility of logically deducing correct settings from an initial state. As an example, in one analysis

| IPS | SCA | Alt | MPS | MSL | LANGUAGE | REF | EPOCHS (syllabic) |
|-----|-----|-----|-----|-----|----------|-----|-------------------|
|  |  | 0 | 0 | 0 | Latvian | *L1* | 17 |
|  |  |  |  |  | French | *L2* | 16 |
|  |  | 0 | 0 | 1 | Latvian2stress | h1 | 21 |
|  |  |  |  |  | Latvian3stress | h2 | 11 |
|  |  |  |  |  | French2stress | h3 | 23 |
|  |  |  |  |  | French3stress | h4 | 14 |
|  |  | 0 | 1 | 0 | Latvian2edge | h5 | 30 |
|  |  | 0 | 1 | 1 | Latvian2edge2stress | h6 | 37 |
|  |  | 1 | 0 | 0 | *impossible* |  |  |
|  |  | 1 | 0 | 1 | Maranungku | *L3* | 37 |
|  |  |  |  |  | Weri | *L4* | 34 |
|  |  |  |  |  | Maranungku3stress | h7 | 43 |
|  |  |  |  |  | Weri3stress | h8 | 41 |
|  |  |  |  |  | Latvian2edge2stress-alt | h9 | 58 |
|  |  |  |  |  | Garawa-SC | h10 | 38 |
|  |  |  |  |  | Garawa2stress-SC | h11 | 50 |
|  |  | 1 | 1 | 0 | Maranungku1stress | h12 | 61 |
|  |  |  |  |  | Weri1stress | h13 | 65 |
|  |  |  |  |  | Latvian2edge-alt | h14 | 78 |
|  |  |  |  |  | Garawa1stress-SC | h15 | 88 |
|  |  | 1 | 1 | 1 | Latvian2edge2stress-1alt | h16 | 85 |
|  | 1 | 0 | 0 | 0 | *impossible* |  |  |
|  | 1 | 0 | 0 | 1 | Garawa-non-alt | h17 | 164 |
|  |  |  |  |  | Latvian3stress2edge-SCA | h18 | 163 |
|  | 1 | 0 | 1 | 0 | Latvian2edge-SCA | h19 | 194 |
|  | 1 | 0 | 1 | 1 | Latvian2edge2stress-SCA | h20 | 206 |
|  | 1 | 1 | 0 | 1 | Garawa | *L5* | 165 |
|  |  |  |  |  | Garawa2stress | h21 | 71 |
|  |  |  |  |  | Latvian2edge2stress-alt-SCA | h22 | 91 |
|  | 1 | 1 | 1 | 0 | Garawa1stress | h23 | 121 |
|  |  |  |  |  | Latvian2edge-alt-SCA | h24 | 126 |
|  | 1 | 1 | 1 | 1 | Latvian2edge2stress-1alt-SCA | h25 | 129 |
| 1 |  | 0 | 0 | 0 | Lakota | *L6* | 255 |
|  |  |  |  |  | Swahili | *L7* | 254 |
| 1 |  | 0 | 0 | 1 | Lakota2stress | h26 | ** |
| 1 |  | 0 | 1 | 0 | Lakota2edge | h27 | ** |
| 1 |  | 0 | 1 | 1 | Lakota2edge2stress | h28 | ** |
| 1 |  | 1 | 0 | 1 | Paiute | *L8* | ** |
|  |  |  |  |  | Warao | *L9* | ** |
| 1 |  | 1 | 1 | 0 | Lakota-alt | h29 | ** |
| 1 |  | 1 | 1 | 1 | Lakota2stress-alt | h30 | ** |

Table 4: Analysis of Quantity-Insensitive learning using the *syllabic* input representation. IPS=Inconsistent Primary Stress; SCA=Stress Clash Avoidance; Alt=Alternation; MPS=Multiple Primary Stresses; MSL=Multiple Stress Levels. References L1-L9 index into Table 1.

[Dresher 90, p. 191], "metrical feet" are taken to be "iterative" by default, since there is evidence that can cause revision of this default if it turns out to be the incorrect setting, but there might not be such disconfirming evidence if the feet were by default taken to be "non-iterative". We provide an alternative to logical deduction arguments for determining "markedness" of parameter values, by measuring learnability (and hence markedness) empirically. The parameters of our novel analysis generate both a partial description of each stress pattern and a prediction of its learnability. Furthermore, our parameters encode linguistically salient concepts (e.g., *stress clash avoidance*) as well as concepts that have computational significance (*single-positional* vs. *aggregative* information.) Although our analyses do not explicitly invoke theoretical linguistic constructs such as metrical feet, there are suggestive similarities between such constructs and the weight patterns the perceptron develops [Gupta 91].

In conclusion, this work offers a fresh perspective on a well-studied linguistic domain, and suggests that machine learning techniques in conjunction with more traditional tools might provide the basis for a new approach to the investigation of language.

## Acknowledgements

We would like to acknowledge the feedback provided by Deirdre Wheeler throughout the course of this work. The first author would like to thank David Evans for access to exceptional computing facilities at Carnegie Mellon's Laboratory for Computational Linguistics, and Dan Everett, Brian MacWhinney, Jay McClelland, Eric Nyberg, Brad Pritchett and Steve Small for helpful discussion of earlier versions of this paper. Of course, none of them is responsible for any errors.

The second author was supported by a grant from Hughes Aircraft Corporation, and by the Office of Naval Research under contract number N00014-86-K-0678.

## Footnotes

[1] For an overview of the theory, see [Goldsmith 90, chapter 4].

[2] See [Dresher 90] for one such parameter scheme.

[3]They were learnable in a three-layer model, which exhibited a similar ordering of learning times [Gupta 92].

[4]Placement of stress on adjacent syllables.

## References

[Dresher 90] Dresher, B., & Kaye, J., A Computational Learning Model for Metrical Phonology, *Cognition* 34, 137-195.

[Goldsmith 90] Goldsmith, J., *Autosegmental and Metrical Phonology*, Basil Blackwell, Oxford, England, 1990.

[Gupta 91] Gupta, P. & Touretzky, D., What a perceptron reveals about metrical phonology. *Proceedings of the Thirteenth Annual Conference of the Cognitive Science Society*, 334-339. Lawrence Erlbaum, Hillsdale, NJ, 1991.

[Gupta 92] Gupta, P. & Touretzky, D., Connectionist Models and Linguistic Theory: Investigations of Stress Systems in Language. Manuscript.

[Hayes 80] Hayes, B., *A Metrical Theory of Stress Rules*, doctoral dissertation, Massachusetts Institute of Technology, Cambridge, MA, 1980. Circulated by the Indiana University Linguistics Club, 1981.

[Rumelhart 86] Rumelhart, D., Hinton, G., & Williams, R, Learning Internal Representations by Error Propagation, in D. Rumelhart, J. McClelland & the PDP Research Group. *Parallel Distributed Processing, Volume 1: Foundations*, MIT Press, Cambridge, MA, 1986.